# How Do Humans Teach:
# On Curriculum Learning and Teaching Dimension

**Faisal Khan,   Xiaojin Zhu,   Bilge Mutlu**
Department of Computer Sciences, University of Wisconsin–Madison
Madison, WI, 53706 USA. {`faisal, jerryzhu, bilge`}`@cs.wisc.edu`

## Abstract

We study the empirical strategies that humans follow as they teach a target concept with a simple 1D threshold to a robot.[1] Previous studies of computational teaching, particularly the teaching dimension model and the curriculum learning principle, offer contradictory predictions on what optimal strategy the teacher should follow in this teaching task. We show through behavioral studies that humans employ three distinct teaching strategies, one of which is consistent with the curriculum learning principle, and propose a novel theoretical framework as a potential explanation for this strategy. This framework, which assumes a teaching goal of minimizing the learner's expected generalization error at each iteration, extends the standard teaching dimension model and offers a theoretical justification for curriculum learning.

## 1   Introduction

With machine learning comes the question of how to effectively *teach*. Computational teaching has been well studied in the machine learning community [9, 12, 10, 1, 2, 11, 13, 18, 4, 14, 15]. However, whether these models can predict how *humans* teach is less clear. The latter question is important not only for such areas as education and cognitive psychology but also for applications of machine learning, as learning agents such as robots become commonplace and learn from humans. A better understanding of the teaching strategies that humans follow might inspire the development of new machine learning models and the design of learning agents that more naturally accommodate these strategies.

Studies of computational teaching have followed two prominent threads. The first thread, developed by the computational learning theory community, is exemplified by the "teaching dimension" model [9] and its extensions [12, 10, 1, 2, 11, 13, 18]. The second thread, motivated partly by observations in psychology [16], is exemplified by the "curriculum learning" principle [4, 14, 15]. We will discuss these two threads in the next section. However, they make conflicting predictions on what optimal strategy a teacher should follow in a simple teaching task. This conflict serves as an opportunity to compare these predictions to human teaching strategies in the same task.

This paper makes two main contributions: (i) it enriches our empirical understanding of human teaching and (ii) it offers a theoretical explanation for a particular teaching strategy humans follow. Our approach combines cognitive psychology and machine learning. We first conduct a behavioral study with human participants in which participants teach a robot, following teaching strategies of their choice. This approach differs from most previous studies of computational teaching in machine learning and psychology that involve a predetermined teaching strategy and that focus on the behavior of the learner rather than the teacher. We then compare the observed human teaching strategies to those predicted by the teaching dimension model and the curriculum learning principle.

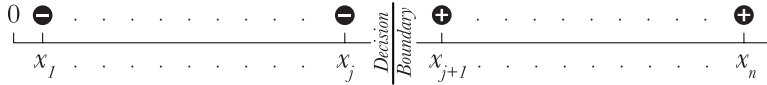

Figure 1: The target concept $h_j$.

Empirical results indicate that human teachers follow the curriculum learning principle, while no evidence of the teaching dimension model is observed. Finally, we provide a novel theoretical analysis that extends recent ideas in teaching dimension model [13, 3] and offers curriculum learning a rigorous underpinning.

## 2 Competing Models of Teaching

We first review the classic teaching dimension model [9, 1]. Let $\mathcal{X}$ be an input space, $\mathcal{Y}$ the label space, and $(x_1, y_1), \ldots, (x_n, y_n) \in \mathcal{X} \times \mathcal{Y}$ a set of instances. We focus on binary classification in the unit interval: $\mathcal{X} = [0, 1], \mathcal{Y} = \{0, 1\}$. We call $\mathcal{H} \subseteq 2^{\{x_1, \ldots, x_n\}}$ a concept class and $h \in H$ a concept. A concept $h$ is consistent with instance $(x, y)$ iff $x \in h \Leftrightarrow y = 1$. $h$ is consistent with a set of instances if it is consistent with every instance in the set. A set of instances is called a teaching set of a concept $h$ with respect to $H$, if $h$ is the only concept in $H$ that is consistent with the set. The teaching dimension of $h$ with respect to $H$ is the minimum size of its teaching set. The teaching dimension of $H$ is the maximum teaching dimension of its concepts.

Consider the task in Figure 1, which we will use throughout the paper. Let $x_1 \leq \ldots \leq x_n$. Let $H$ be all threshold labelings: $H = \{h \mid \exists \theta \in [0, 1], \forall i = 1 \ldots n : x_i \in h \Leftrightarrow x_i \geq \theta\}$. The target concept $h_j$ has the threshold between $x_j$ and $x_{j+1}$: $h_j = \{x_{j+1}, \ldots, x_n\}$. Then, the teaching dimension of most $h_j$ is 2, as one needs the minimum teaching set $\{(x_j, 0), (x_{j+1}, 1)\}$; for the special cases $h_0 = \{x_1, \ldots, x_n\}$ and $h_n = \emptyset$ the teaching dimension is 1 with the teaching set $\{(x_1, 1)\}$ and $\{(x_n, 0)\}$, respectively. The teaching dimension of $H$ is 2. For our purpose, the most important argument is the following: *The teaching strategy for most $h_j$'s suggested by teaching dimension is to show two instances $\{(x_j, 0), (x_{j+1}, 1)\}$ closest to the decision boundary.* Intuitively, these are the instances most confusable by the learner.

Alternatively, curriculum learning suggests an easy-to-hard (or clear-to-ambiguous) teaching strategy [4]. For the target concept in Figure 1, "easy" instances are those farthest from the decision boundary in each class, while "hard" ones are the closest to the boundary. *One such teaching strategy is to present instances from alternating classes, e.g., in the following order:* $(x_1, 0), (x_n, 1), (x_2, 0), (x_{n-1}, 1), \ldots, (x_j, 0), (x_{j+1}, 1)$. Such a strategy has been used for second-language teaching in humans. For example, to train Japanese listeners on the English [r]-[l] distinction, McCandliss *et al.* linearly interpolated a vocal tract model to create a 1D continuum similar to Figure 1 along [r] and [l] sounds. They showed that participants were better able to distinguish the two phonemes if they were given easy (over-articulated) training instances first [16]. Computationally, curriculum learning has been justified as a heuristic related to continuation method in optimization to avoid poor local optima [4].

Hence, for the task in Figure 1, we have two sharply contrasting teaching strategies at hand: the `boundary` strategy starts near the decision boundary, while the `extreme` strategy starts with extreme instances and gradually approaches the decision boundary from both sides. Our goal in this paper is to compare human teaching strategies with these two predictions to shed more light on models of teaching. While the teaching task used in our exploration is simple, as most real-world teaching situations do not involve a threshold in a 1D space, we believe that it is important to lay the foundation in a tractable task before studying more complex tasks.

## 3 A Human Teaching Behavioral Study

Under IRB approval, we conducted a behavioral study with human participants to explore human teaching behaviors in a task similar to that illustrated in Figure 1. In our study, participants teach the target concept of "graspability"—whether an object can be grasped and picked up with one hand—to a robot. We chose graspability because it corresponds nicely to a 1D space empirically

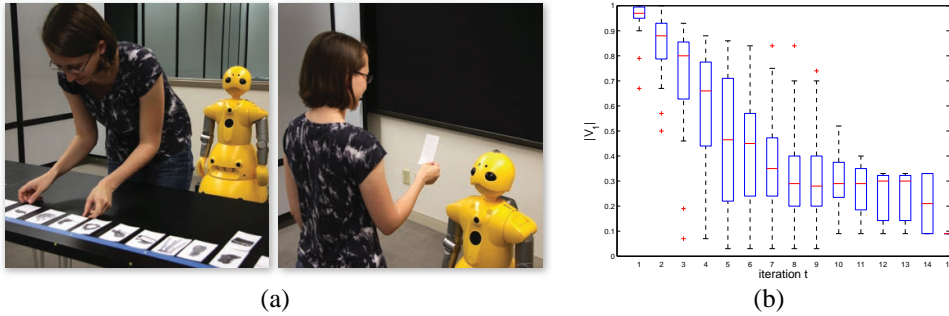

(a)                                        (b)

Figure 2: (a) A participant performing the card sorting/labeling and teaching tasks. (b) Human teaching sequences that follow the `extreme` strategy gradually shrink the version space $V_1$.

studied before [17]. We chose to use a robot learner because it offers great control and consistency while facilitating natural interaction and teaching. The robot keeps its behavior consistent across conditions and trials, therefore, providing us with the ability to isolate various interactional factors. This level of experimental control is hard to achieve with a human learner. The robot also affords embodied behavioral cues that facilitate natural interaction and teaching strategies that computers do not afford.

**Participants** were 31 paid subjects recruited from the University of Wisconsin–Madison campus. All were native English speakers with an average age of 21 years.

**Materials.** We used black-and-white photos of $n = 31$ objects chosen from the norming study of Salmon *et al.* [17]. The photos were of common objects (e.g., food, furniture, animals) whose average subjective graspability ratings evenly span the whole range. We printed each photo on a 2.5-by-4.5 inch card. The robot was a Wakamaru humanlike robot manufactured by Mitsubishi Heavy Industries, Ltd. It neither learned nor responded to teaching. Instead, it was programmed to follow motion in the room with its gaze. Though seemingly senseless, this behavior in fact provides a consistent experience to the participants without extraneous factors to bias them. It also corresponds to the no-feedback assumption in most teaching models [3]. Participants were not informed that the robot was not actually learning.

**Procedure.** Each participant completed the experiment alone. The experiment involved two sub-tasks that were further broken down into multiple steps. In the first subtask, participants sorted the objects based on their subjective ratings of their graspability following the steps below.

In step 1, participants were instructed to place each object along a ruler provided on a long table as seen in Figure 2(a). To provide baselines on the two ends of the graspability spectrum, we fixed a highly graspable object (a toothbrush) and a highly non-graspable object (a building) on the two ends of the ruler. We captured the image of the table and later converted the position of each card into a participant-specific, continuous graspability rating $x_1, \ldots, x_n \in [0, 1]$. For our purpose, there is no need to enforce inter-participant agreement.

In step 2, participants assigned a binary "graspable" ($y = 1$) or "not graspable" ($y = 0$) label to each object by writing the label on the back of the corresponding card. This gave us labels $y_1, \ldots, y_n$. The sorted cards and the decision boundary from one of the participants is illustrated in Figure 3.

In step 3, we asked participants to leave the room for a short duration so that "the robot could examine the sorted cards on the table without looking at the labels provided at the back," creating the impression that the learner will associate the cards with the corresponding values $x_1, \ldots, x_n$.

In the second subtask, participants taught the robot the (binary) concept of graspability using the cards. In this task, participants picked up a card from the table, turned toward the robot, and held the card up while providing a verbal description of the object's graspability (i.e., the binary label $y$) as seen in Figure 2(a). The two cards, "toothbrush" and "building," were fixed to the table and not available for teaching. The participants were randomly assigned into two conditions: (1) natural and (2) constrained. In the "natural" condition, participants were allowed to use natural language to describe the graspability of the objects, while those in the "constrained" condition were only allowed

to say either "graspable" or "not graspable." They were instructed to use as *few* cards as they felt necessary. There was no time limit on either subtasks.

**Results.** The teaching sequences from all participants are presented in Figure 4. The title of each plot contains the participant ID and condition. The participant's rating and classification of all objects are presented above the $x$-axis. Objects labeled as "not graspable" are indicated with blue circles and those labeled as "graspable" are marked with red plus signs. The $x$-axis position of the object represents its rating $x \in [0, 1]$. The vertical blue and red lines denote an "ambiguous region" around the decision boundary; objects to the left of the blue line have the label "not graspable;" those to the right of the red line are labeled as "graspable," and objects between these lines *could* have labels in mixed order. In theory, following the `boundary` strategy, the teacher should start with teaching instances on these two lines as suggested by the teaching dimension model. The $y$-axis is trial $t = 1, \ldots, 15$, which progresses upwards. The black line and dots represent the participant's teaching sequence. For example, participant P01 started teaching at $t = 1$ with an object she rated as $x = 1$ and labeled as "graspable;" at $t = 2$, she chose an example with rating $x = 0$ and label "not graspable;" and so on. The average teaching sequence had approximately 8 examples, while the longest teaching sequence had a length of 15 examples.

We observed *three* major human teaching strategies in our data: (1) the `extreme` strategy, which starts with objects with extreme ratings and gradually moves toward the decision boundary; (2) the `linear` strategy, which follows a prominent left-to-right or right-to-left sequence; and (3) the `positive-only` strategy, which involves only positively labeled examples. We categorized most teaching sequences into these three strategies following a simple heuristic. First, sequences that involved only positive examples were assigned to the `positive-only` strategy. Then, we assigned the sequences whose first two teaching examples had different labels to the `extreme` strategy and the others to the `linear` strategy. While this simplistic approach does not guarantee perfect classification (e.g., P30 can be labeled differently), it minimizes hand-tuning and reduces the risk of overfitting. We made two exceptions, manually assigning P14 and P16 to the extreme strategy. Nonetheless, these few potential misclassifications do not change our conclusions below.

None of the sequences followed the `boundary` strategy. In fact, among all 31 participants, 20 started teaching with the most graspable object (according to their own rating), 6 with the least graspable, none in or around the ambiguous region (as `boundary` strategy would predict), and 5 with some other objects. In brief, people showed a tendency to start teaching with extreme objects, especially the most graspable ones. During post-interview, when asked why they did not start with objects around their decision boundary, most participants mentioned that they wanted to start with *clear* examples of graspability.

For participants who followed the `extreme` strategy, we are interested in whether their teaching sequences approach the decision boundary as curriculum learning predicts. Specifically, at any time $t$, let the partial teaching sequence be $(x_1, y_1), \ldots, (x_t, y_t)$. The aforementioned ambiguous region with respect to this partial sequence is the interval between the inner-most pair of teaching examples with different labels. This can be written as $V_1 \equiv [\max_{j:y_j=0} x_j, \min_{j:y_j=1} x_j]$ where $j$ is over $1 \ldots t$. $V_1$ is exactly the *version space* of consistent threshold hypotheses (the subscript 1 will become clear in the next section). Figure 2(b) shows a box plot of the size of $V_1$ for all participants as a function of $t$. The red lines mark the median and the blue boxes indicate the 1st & 3rd quartiles. As expected, the size of the version space decreases.

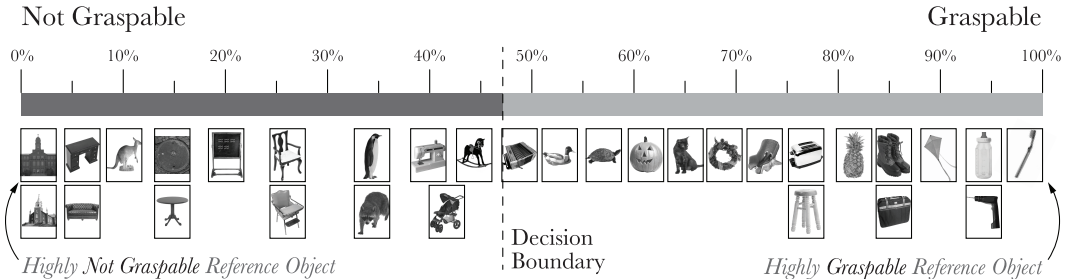

Figure 3: Sorted cards and the decision boundary from one of the participants.

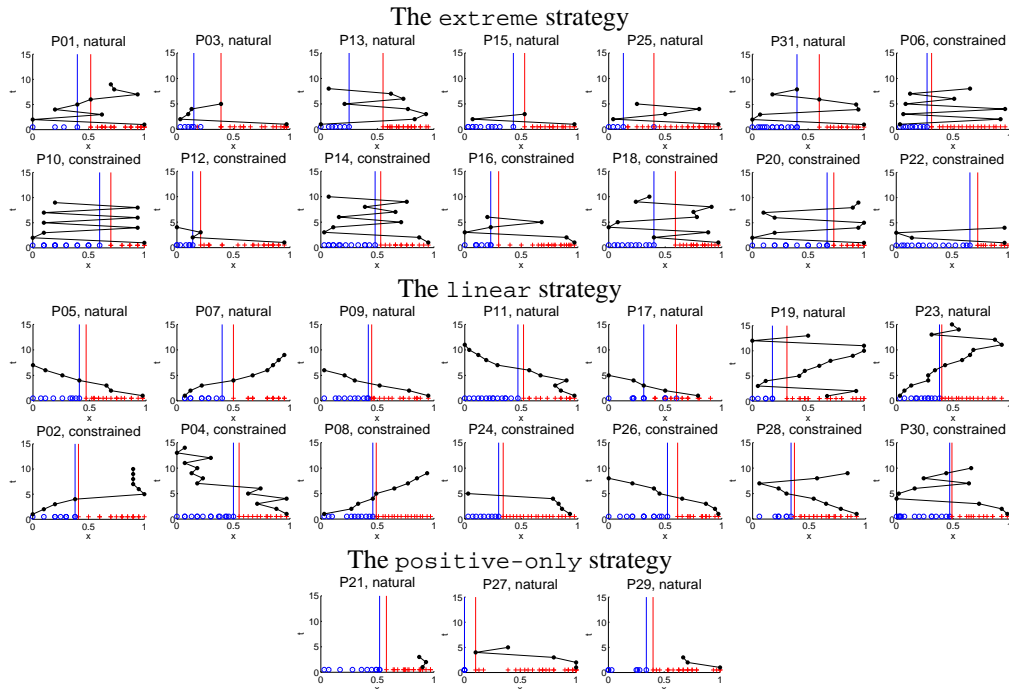

Figure 4: Teaching sequences of all participants.

Finally, the `positive-only` strategy was observed significantly more in the "natural" condition ($3/16 \approx 19\%$) than in the "constrained" condition ($0/15 = 0\%$), $\chi^2(1, N = 31) = 4.27, p = .04$. We observed that these participants elaborated in English to the robot why they thought that their objects were graspable. We speculate that they might have felt that they had successfully described the *rules* and that there was no need to use negative examples. In contrast, the constrained condition did not have the rich expressivity of natural language, necessitating the use of negative examples.

## 4   A Theoretical Account of the "Extreme" Teaching Strategy

We build on our empirical results and offer a theoretical analysis as a possible rationalization for the `extreme` strategy. Research in cognitive psychology has consistently shown that humans represent everyday objects with a large number of features (e.g., [7, 8]). We posit that although our teaching task was designed to mimic the one-dimensional task illustrated in Figure 1 (e.g., the linear layout of the cards in Figure 3), our teachers might still have believed (perhaps subconsciously) that the robot learner, like humans, associates each teaching object with multiple feature dimensions.

Under the high-dimensional assumption, we show that the `extreme` strategy is an outcome of minimizing per-iteration expected error of the learner. Note that the classic teaching dimension model [9] fails to predict the `extreme` strategy even under this assumption. Our analysis is inspired by recent advances in teaching dimension, which assume that teaching progresses in iterations and learning is to be maximized after each iteration [13, 3]. Different from those analysis, we minimize the *expected error* instead of the *worst-case error* and employ different techniques.

### 4.1   Problem Setting and Model Assumptions

Our formal set up is as follows. The instance space is the $d$-dimensional hypercube $\mathcal{X} = [0, 1]^d$. We use boldface $\mathbf{x} \in \mathcal{X}$ to denote an instance and $x_{ij}$ for the $j$-th dimension of instance $\mathbf{x}_i$. The binary label $y$ is determined by the threshold $\frac{1}{2}$ in the first dimension: $y_i = \mathbf{1}_{\{x_{i1} \geq \frac{1}{2}\}}$. This formulation idealizes our empirical study where the continuous rating is the first dimension. It implies that the target concept is unrelated to any of the other $d - 1$ features. In practice, however, there may be other

features that are correlated with the target concept. But our analysis carries through by replacing $d$ with the number of irrelevant dimensions.

Departing from classic teaching models, we consider a "pool-based sequential" teaching setting. In this setting, a pool of $n$ instances are sampled *iid* $\mathbf{x}_1, \ldots, \mathbf{x}_n \sim p(\mathbf{x})$, where we assume that $p(\mathbf{x})$ is uniform on $\mathcal{X}$ for simplicity. Their labels $y_1 \ldots y_n$ may be viewed as being sampled from the conditional distribution $p(y_i = 1 \mid \mathbf{x}_i) = \mathbf{1}_{\{x_{i1} > \frac{1}{2}\}}$. The teacher can only sequentially teach instances selected from the pool (e.g., in our empirical study, the pool consists of the 29 objects). Her goal is for the learner to generalize well on test instances outside the pool (also sampled from $p(\mathbf{x}, y) = p(\mathbf{x})p(y \mid \mathbf{x})$) after each iteration.

At this point, we make two strong assumptions on the learner. First, we assume that the learner entertains axis-parallel hypotheses. That is, each hypothesis has the form $h_{k\theta s}(\mathbf{x}) = \mathbf{1}_{\{s(x_{\cdot k} - \theta) \geq 0\}}$ for some dimension $k \in \{1 \ldots d\}$, threshold $\theta \in [0, 1]$, and orientation $s \in \{-1, 1\}$. The cognitive interpretation of an axis-parallel hypothesis is that the learner attends to a single dimension at any given time.[2] As in classic teaching models, our learner is consistent (i.e., it never contradicts with the teaching instances it receives). The *version space* $V(t)$ of the learner, i.e., the set of hypotheses that is consistent with the teaching sequence $(\mathbf{x}_1, y_1), \ldots, (\mathbf{x}_t, y_t)$ so far, takes the form $V(t) = \cup_{k=1}^{d} V_k(t)$ where $V_k(t) = \{h_{k\theta,1} \mid \max_{j:y_j=0} x_{jk} \leq \theta \leq \min_{j:y_j=1} x_{jk}\} \cup \{h_{k\theta,-1} \mid \max_{j:y_j=1} x_{jk} \leq \theta \leq \min_{j:y_j=0} x_{jk}\}$. The version space can be thought of as the union of inner intervals surviving the teaching examples.

Second, similar to the randomized learners in [2], our learner selects a hypothesis $h$ uniformly from the version space $V(t)$, follows it until when $h$ is no longer in $V(t)$, and then randomly selects a replacement hypothesis—a strategy known as "win stay, lose shift" in cognitive psychology [5]. It is thus a Gibbs classifier. In particular, the risk, defined as the expected 0-1 loss of the learner on a test instance, is $R(t) \equiv \mathbb{E}_{(\mathbf{x},y) \sim p(\mathbf{x},y)} \mathbb{E}_{h \in V(t)} \mathbf{1}_{\{h(\mathbf{x}) \neq y\}}$. We point out that our assumptions are psychologically plausible and will greatly simplify the derivation below.

## 4.2 Starting with Extreme Teaching Instances is Asymptotically Optimal

We now show why starting with extreme teaching instances as in curriculum learning, as opposed to the `boundary` strategy, is optimal under our setting. Specifically, we consider the problem of selecting an optimal teaching sequence of length $t = 2$, one positive and one negative, $(\mathbf{x}_1, 1), (\mathbf{x}_2, 0)$. Introducing the shorthand $a \equiv x_{11}, b \equiv x_{21}$, the teacher seeks $a, b$ to minimize the risk:

$$\min_{a,b \in [0,1]} R(2) \tag{1}$$

Note that we allow $a, b$ to take any value within their domains, which is equivalent to having an infinite pool for the teacher to choose from. We will tighten it later. Also note that we assume the teacher does not pay attention to irrelevant dimensions, whose feature values can then be modeled by uniform random variables.

For any teaching sequence of length 2, the individual intervals of the version space are of size $|V_1(2)| = a - b$, $|V_k(2)| = |x_{1k} - x_{2k}|$ for $k = 2 \ldots d$, respectively. The total size of the version space is $|V(2)| = a - b + \sum_{k=2}^{d} |x_{1k} - x_{2k}|$. Figure 5(a) shows that for all $h_{1\theta_1 1} \in V_1(2)$, the decision boundary is parallel to the true decision boundary and the test error is $\mathbb{E}_{(\mathbf{x},y) \sim p(\mathbf{x},y)} \mathbf{1}_{\{h_{1\theta_1 1}(\mathbf{x}) \neq y\}} = |\theta_1 - 1/2|$. Figure 5(b) shows that for all $h_{k\theta_k s} \in \cup_{k=2}^{d} V_k(2)$, the decision boundary is orthogonal to the true decision boundary and the test error is $1/2$. Therefore, we have $R(2) = \frac{1}{|V(2)|} \left( \int_b^a |\theta_1 - 1/2| d\theta_1 + \sum_{k=2}^{d} \int_{\min(x_{1k}, x_{2k})}^{\max(x_{1k}, x_{2k})} \frac{1}{2} d\theta_k \right) = \frac{1}{|V(2)|} \left( \frac{1}{2}(\frac{1}{2} - b)^2 + \frac{1}{2}(a - \frac{1}{2})^2 + \sum_{k=2}^{d} \frac{1}{2} |x_{1k} - x_{2k}| \right)$. Introducing the shorthand $c_k \equiv |x_{1k} - x_{2k}|$, $c \equiv \sum_{k=2}^{d} c_k$, one can write $R(2) = \frac{(\frac{1}{2} - b)^2 + (a - \frac{1}{2})^2 + c}{2(a - b + c)}$. The intuition is that a pair of teaching instances lead to a version space $V(2)$ consisting of one interval per dimension. A random hypothesis selected from the interval in the first dimension $V_1(2)$ can range from good (if $\theta_1$ is close

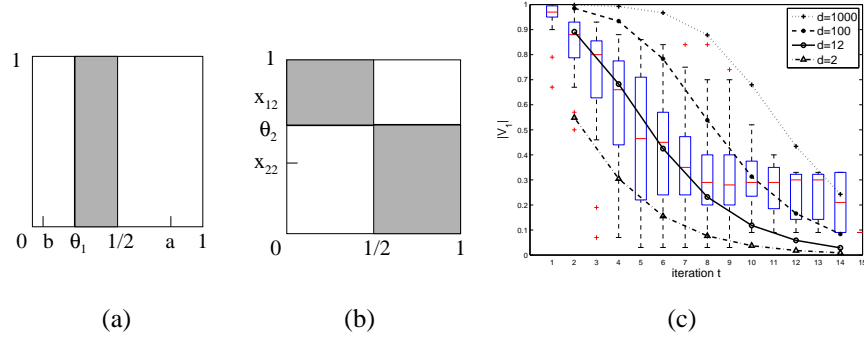

(a)                      (b)                      (c)

Figure 5: (a) A hypothesis $h_{1\theta_1 1} \in V_1(2)$ is parallel to the true decision boundary, with test error $|\theta_1 - 1/2|$ (shaded area). (b) A hypothesis $h_{2\theta_2 s} \in V_2(2)$ is orthogonal to the true decision boundary, with test error $1/2$ (shaded area). (c) Theoretical teaching sequences gradually shrink $|V_1|$, similar to human behaviors.

to 1/2) to poor ($\theta_1$ far away from 1/2), while one selected from $\cup_{k=2}^d V_k(2)$ is always bad. The teacher can optimize the risk by choosing the size of $V_1(T)$ related to the total version space size. The optimal choice is specified by the following theorem.

**Theorem 1.** *The minimum risk $R(2)$ is achieved at $a = \frac{\sqrt{c^2+2c}-c+1}{2}$, $b = 1 - a$.*

*Proof.* First, we show that at the minimum $a, b$ are symmetric around $1/2$, i.e., $b = 1 - a$. Suppose not. Then, $(a+b)/2 = 1/2+\epsilon$ for some $\epsilon \neq 0$. Let $a' = a-\epsilon, b' = b-\epsilon$. Then, $\frac{(\frac{1}{2}-b')^2+(a'-\frac{1}{2})^2+c}{2(a'-b'+c)} = \frac{(\frac{1}{2}-b)^2+(a-\frac{1}{2})^2+c-2\epsilon^2}{2(a-b+c)} < \frac{(\frac{1}{2}-b)^2+(a-\frac{1}{2})^2+c}{2(a-b+c)}$ the minimum, a contradiction. Next, substituting $b = 1 - a$ in $R(2)$ and setting the derivative w.r.t. $a$ to 0 proves the theorem. $\qquad\square$

Recall that $c$ is the size of the part of the version space in irrelevant dimensions. When $d \to \infty$, $c \to \infty$ and the solution is $a = 1, b = 0$. Here, the learner can form so many bad hypotheses in the many wrong dimensions that the best strategy for the teacher is to make $V_1(2)$ as large as possible, even though many hypotheses in $V_1(2)$ have nonzero error.

**Corollary 2.** *The minimizer to (1) is $a = 1, b = 0$ when the dimensionality $d \to \infty$.*

*Proof.* We characterize the distribution of $c_k$ by considering the distance between two random variables $x_{1k}, x_{2k}$ sampled uniformly in $[0, 1]$. Let $z_{(1)}, z_{(2)}$ be the values of $x_{1k}, x_{2k}$ sorted in an ascending order. Then $c_k = z_{(2)} - z_{(1)}$ is an instance of *order statistics* [6]. One can show that, in general with $t$ independent $\mathrm{unif}[0, 1]$ random variables sorted in an ascending order as $z_{(1)}, \ldots, z_{(j)}, z_{(j+1)}, \ldots, z_{(t)}$, the distance $z_{(j+1)} - z_{(j)}$ follows a $\mathrm{Beta}(1, t)$ distribution. In our case with $t = 2$, $c_k \sim \mathrm{Beta}(1, 2)$, whose mean is $1/3$ as expected. It follows that $c$ is the sum of $d - 1$ independent Beta random variables. As $d \to \infty$, $c \to \infty$. Let $\gamma = 1/c$. Applying l'Hôpital's rule, $\lim_{c\to\infty} a = \lim_{c\to\infty} \frac{\sqrt{c^2+2c}-c+1}{2} = \lim_{\gamma\to 0} \frac{\sqrt{1+2\gamma}-1+\gamma}{2\gamma} = 1$. $\qquad\square$

Corollary 2 has an interesting cognitive interpretation; the teacher only needs to pay attention to the relevant (first) dimension $x_{11}, x_{21}$ when selecting the two teaching instances. She does not need to consider the irrelevant dimensions, as those will add up to a large $c$, which simplifies the teacher's task in choosing a teaching sequence; she simply picks two extreme instances in the first dimension. We also note that in practice $d$ does not need to be very large for $a$ to be close to 1. For example, with $d = 10$ dimensions, the average $c$ is $\frac{1}{3}(d-1) = 3$ and the corresponding $a = 0.94$, with $d = 100$, $a = 0.99$. This observation provides further psychological plausibility to our model.

So far, we have assumed an infinite pool, such that the teacher can select the extreme teaching instances with $x_{11} = 1, x_{21} = 0$. In practice, the pool is finite and the optimal $a, b$ values specified in Theorem 1 may not be attainable within the pool. However, it is straightforward to show that $\lim_{c\to\infty} R'(t) < 0$ where the derivative is w.r.t. $a$ after substituting $b = 1 - a$. That is, in the case of $c \to \infty$, the objective in (1) is a monotonically decreasing function of $a$. Therefore, the optimal strategy for a finite pool is to choose the negative instance with the smallest $x_{\cdot 1}$ value and

the positive instance with the largest $x_{\cdot 1}$ value. Note the similarity to curriculum learning which starts with extreme (easy) instances.

## 4.3 The Teaching Sequence should Gradually Approach the Boundary

Thus far, we have focused on choosing the first two teaching instances. We now show that, as teaching continues, the teacher should choose instances with $a$ and $b$ gradually approaching $1/2$. This is a direct consequence of minimizing the risk $R(t)$ at each iteration, as $c$ decreases to 0. In this section, we study the speed by which $c$ decreases to 0 and $a$ to 1/2.

Consider the moment when the teacher has already presented a teaching sequence $(\mathbf{x}_1, y_1), \ldots, (\mathbf{x}_{t-2}, y_{t-2})$ and is about to select the next pair of teaching instances $(\mathbf{x}_{t-1}, 1), (\mathbf{x}_t, 0)$. Teaching with pairs is not crucial but will simplify the analysis. Following the discussion after Corollary 2, we assume that the teacher only pays attention to the first dimension when selecting teaching instances. This assumption allows us to again model the other dimensions as random variables. The teacher wishes to determine the optimal $a = x_{t-1,1}, b = x_{t,1}$ values according to Theorem 1. What is the value of $c$ for a teaching sequence of length $t$?

**Theorem 3.** *Let the teaching sequence contain $t_0$ negative labels and $t - t_0$ positive ones. Then the random variables $c_k = \alpha_k \beta_k$, where $\alpha_k \sim \text{Bernoulli}\left(2/\binom{t}{t_0}, 1 - 2/\binom{t}{t_0}\right)$ (with values 1, 0 respectively) and $\beta_k \sim \text{Beta}(1, t)$ independently for $k = 2 \ldots d$. Consequently, $\mathbb{E}(c) = \frac{2(d-1)}{\binom{t}{t_0}(1+t)}$.*

*Proof.* We show that for each irrelevant dimension $k = 2 \ldots d$, after $t$ teaching instances, $|V_k(t)| = \alpha_k \beta_k$. As mentioned above, these $t$ teaching instances can be viewed as $\text{unif}[0, 1]$ random variables in the $k$th dimension. Sort the values $x_{1k}, \ldots, x_{tk}$ in ascending order. Denote the sorted values as $z_{(1)}, \ldots, z_{(t)}$. $V_k(t)$ is non-empty only if the labels happen to be linearly separable, i.e., either $z_{(1)} \ldots z_{(t_0)}$ having negative labels while the rest having positive labels or the other way around. Consider the corresponding analogy where one randomly selects a permutation of $t$ items (there are $t!$ permutations), such that the selected permutation has first $t_0$ items with negative labels and the rest with positive labels (there are $t_0!(t - t_0)!$ such permutations). This probability corresponds to $\alpha_k$. When $V_k(t)$ is nonempty, its size $|V_k(t)|$ is characterized by the order statistics $z_{(t_0+1)} - z_{(t_0)}$, which corresponds to the Beta random variable $\beta_k$ as mentioned earlier in the proof of Corollary 2. $\qquad\square$

As the binomial coefficient in the denominator of $\mathbb{E}(c)$ suggests, $c$ decreases to 0 rapidly with $t$, because $t$ randomly-placed labels in 1D are increasingly unlikely to be linearly separable. Following Theorem 1, the corresponding optimal $a, b$ approach 1/2. Due to the form of Theorem 1, the pace is slower. To illustrate how fast the optimal teaching sequence approaches 1/2 in the first dimension, Figure 5(c) shows a plot of $|V_1| = a - b$ as a function of $t$ by using $\mathbb{E}(c)$ in Theorem 1 (note in general that this is not $\mathbb{E}(|V_1|)$, but only a typical value). We set $t_0 = t/2$. This plot is similar to the one we produced from human behavioral data in Figure 2(b). For comparison, that plot is copied here in the background. Because the effective number of independent dimensions $d$ is unknown, we present several curves for different $d$'s. Some of these curves provide a qualitatively reasonable fit to human behavior, despite the fact that we made several simplifying model assumptions.

## 5 Conclusion and Future Work

We conducted a human teaching experiment and observed three distinct human teaching strategies. Empirical results yielded no evidence for the `boundary` strategy but showed that the `extreme` strategy is consistent with the curriculum learning principle. We presented a theoretical framework that extends teaching dimension and explains two defining properties of the `extreme` strategy: (1) teaching starts with extreme instances and (2) teaching gradually approaches the decision boundary.

Our framework predicts that, in the absence of irrelevant dimensions ($d = 1$), teaching should start at the decision boundary. To verify this prediction, in our future work, we plan to conduct additional human teaching studies where the objects have no irrelevant attributes. We also plan to further investigate and explain the `linear` strategy and the `positive-only` strategy that we observed in our current study.

**Acknowledgments:** We thank Li Zhang and Eftychios Sifakis for helpful comments. Research supported by NSF IIS-0953219, IIS-0916038, AFOSR FA9550-09-1-0313, Wisconsin Alumni Research Foundation, and Mitsubishi Heavy Industries, Ltd.

## Footnotes

[1] Our data is available at `http://pages.cs.wisc.edu/~jerryzhu/pub/humanteaching.tgz`.

[2] A generalization to arbitrary non-axis parallel linear separators is possible in theory and would be interesting. However, non-axis parallel linear separators (known as "information integration" in psychology) are more challenging for human learners. Consequently, our human *teachers* might not have expected the robot learner to perform information integration either.

# References

[1] D. Angluin. Queries revisited. *Theoretical Computer Science*, 313(2):175–194, 2004.

[2] F. J. Balbach and T. Zeugmann. Teaching randomized learners. In *Proceedings of the 19th Annual Conference on Computational Learning Theory (COLT)*, pages 229–243. Springer, 2006.

[3] F. J. Balbach and T. Zeugmann. Recent developments in algorithmic teaching. In *Proceedings of the 3rd International Conference on Language and Automata Theory and Applications*, pages 1–18, 2009.

[4] Y. Bengio, J. Louradour, R. Collobert, and J. Weston. Curriculum learning. In L. Bottou and M. Littman, editors, *Proceedings of the 26th International Conference on Machine Learning*, pages 41–48, Montreal, June 2009. Omnipress.

[5] J. S. Bruner, J. J. Goodnow, and G. A. Austin. *A Study of Thinking*. New York: Wiley, 1956.

[6] H. A. David and H. N. Nagaraja. *Order Statistics*. Wiley, 3rd edition, 2003.

[7] S. De Deyne and G. Storms. Word associations: Network and semantic properties. *Behavior Research Methods*, 40:213–231, 2008.

[8] S. De Deyne and G. Storms. Word associations: Norms for 1,424 Dutch words in a continuous task. *Behavior Research Methods*, 40:198–205, 2008.

[9] S. Goldman and M. Kearns. On the complexity of teaching. *Journal of Computer and Systems Sciences*, 50(1):20–31, 1995.

[10] S. Goldman and H. Mathias. Teaching a smarter learner. *Journal of Computer and Systems Sciences*, 52(2):255267, 1996.

[11] S. Hanneke. Teaching dimension and the complexity of active learning. In *Proceedings of the 20th Annual Conference on Computational Learning Theory (COLT)*, page 6681, 2007.

[12] T. Hegedüs. Generalized teaching dimensions and the query complexity of learning. In *Proceedings of the eighth Annual Conference on Computational Learning Theory (COLT)*, pages 108–117, 1995.

[13] H. Kobayashi and A. Shinohara. Complexity of teaching by a restricted number of examples. In *Proceedings of the 22nd Annual Conference on Computational Learning Theory (COLT)*, pages 293–302, 2009.

[14] M. P. Kumar, B. Packer, and D. Koller. Self-paced learning for latent variable models. In *NIPS*, 2010.

[15] Y. J. Lee and K. Grauman. Learning the easy things first: Self-paced visual category discovery. In *Proceedings of the IEEE Conference on Computer Vision and Pattern Recognition (CVPR)*, 2011.

[16] B. D. McCandliss, J. A. Fiez, A. Protopapas, M. Conway, and J. L. McClelland. Success and failure in teaching the [r]-[l] contrast to Japanese adults: Tests of a Hebbian model of plasticity and stabilization in spoken language perception. *Cognitive, Affective, & Behavioral Neuroscience*, 2(2):89–108, 2002.

[17] J. P. Salmon, P. A. McMullen, and J. H. Filliter. Norms for two types of manipulability (graspability and functional usage), familiarity, and age of acquisition for 320 photographs of objects. *Behavior Research Methods*, 42(1):82–95, 2010.

[18] S. Zilles, S. Lange, R. Holte, and M. Zinkevich. Models of cooperative teaching and learning. *Journal of Machine Learning Research*, 12:349–384, 2011.

